# Spatial Decorrelation in Orientation Tuned Cortical Cells

**Alexander Dimitrov**
Department of Mathematics
University of Chicago
Chicago, IL 60637
a-dimitrov@uchicago.edu

**Jack D. Cowan**
Department of Mathematics
University of Chicago
Chicago, IL 60637
cowan@math.uchicago.edu

## Abstract

In this paper we propose a model for the lateral connectivity of orientation-selective cells in the visual cortex based on information-theoretic considerations. We study the properties of the input signal to the visual cortex and find new statistical structures which have not been processed in the retino-geniculate pathway. Applying the idea that the system optimizes the representation of incoming signals, we derive the lateral connectivity that will achieve this for a set of local orientation-selective patches, as well as the complete spatial structure of a layer of such patches. We compare the results with various physiological measurements.

## 1 Introduction

In recent years much work has been done on how the structure of the visual system reflects properties of the visual environment (Atick and Redlich 1992; Attneave 1954; Barlow 1989). Based on the statistics of natural scenes compiled and studied by Field (1987) and Ruderman and Bialek (1993), work was done by Atick and Redlich (1992) on the assumption that one of the tasks of early vision is to reduce the redundancy of input signals, the results of which agree qualitatively with numerous physiological and psychophysical experiments. Their ideas were further strengthened by research suggesting the possibility that such structures develop via simple correlation-based learning mechanisms (Atick and Redlich 1993; Dong 1994).

As suggested by Atick and Li (1994), further higher-order redundancy reduction of the luminosity field in the visual processing system is unlikely, since it gives little benefit in information compression. In this paper we apply similar ideas to a different input signal which is readily available to the system and whose statistical properties are lost in the analysis of the luminosity signal. We note that after the

application of the retinal "mexican hat" filter the most obvious salient features that are left in images are sharp changes in luminosity, for which the filter is not optimal, i.e. local edges. Such edges have correlations which are very different from the luminosity autocorrelation of natural images (Field 1987), and have zero probability measure in visual scenes, so they are lost in the ensemble averages used to compute the autocorrelation function of natural images. We know that this signal is projected to a set of direction-sensitive units in V1 for each distinct retinal position, thereby introducing new redundancy in the signal. Thus the necessity for compression and use of factorial codes arises once again.

Since local edges are defined by sharp changes in the luminosity field, we can use a derivative operation to pick up the pertinent structure. Indeed, if we look at the gradient of the luminosity as a vector field, its magnitude at a point is proportional to the change of luminosity, so that a large magnitude signals the possible presence of a discontinuity in the luminosity profile. Moreover, in two dimensions, the direction of the gradient vector is perpendicular to the direction of the possible local edge, whose presence is given by the magnitude. These properties define a one-to-one correspondence between large gradients and local edges.

The structure of the network we use reflects what is known about the structure of V1. We select as our system a layer of direction sensitive cells which are laterally connected to one another, each receiving input from the previous layer. We assume that each unit receives as input the directional derivative of the luminosity signal along the preferred visuotopic axis of the cell. This implies that locally the input to a cell is proportional to the cosine of the angle between the unit's preferred direction and the local gradient (edge). Thus each unit receives a broadly tuned signal, with HW-HH approximately 60°. With this feed-forward structure, the idea that the system is trying to decorrelate its inputs suggests a way to calculate the lateral connections that will perform this task. This calculation, and a further study of the statistical properties of the input is the topic of the paper.

## 2   Mathematical Model

Let $G(x) = (G_1(x), G_2(x))$ be the gradient of luminosity at $x$. Assume that there is a set of $N$ detectors with activity $O_i$ at $x$, each with a preferred direction $n_i$. Let the input from the previous layer to each detector be the directional derivative along its preferred direction.

$$V_i(x) = |Grad(L(x)).n_i| = |\frac{d}{dn_i}L(x)| \qquad (1)$$

There are long range correlations in the inputs to the network due both to the statistical structure of the natural images and the structure of the input. The simplest of them are captured in the two-point correlation matrix $R_{ij}(x_1, x_2) = < V_i(x_1)V_j(x_2) >$, where the averaging is done across images. Then $R$ is a block matrix, with an $N \times N$ matrix at each spatial position $(x_1, x_2)$.

We formulate the problem in terms of a recurrent kernel $W$, so that

$$O = V + W * O \qquad (2)$$

The biological interpretation of this is that $V$ is the effective input to V1 from the LGN and $W$ specifies the lateral connectivity in V1. This equation describes the steady state of the linear dynamical system $\dot{O} = -O + W * O + V$. The

above recurrent system has a solution for $O$ not an eigenfunction of $W$ in the form $O = (\delta - W)^{-1} * V = K * V$. This suggests that there is an equivalent feed-forward system with a transfer function $K = (\delta - W)^{-1}$ and we can consider only such systems.

The corresponding feed-forward system is a linear system that acts on the input $V(x)$ to produce an output $O(x) = (K \cdot V)(x) \equiv \int K(x, y) \cdot V(y) dy$. If we use Barlow's redundancy reduction hypothesis (Barlow 1989), this filter should decorrelate the output signal. This is achieved by requiring that

$$
\begin{aligned}
\delta(x_1 - x_2) &\sim & < O(x_1) \circ O(x_2) >=< (K \cdot V)(x_1) \circ (K \cdot V)(x_2) > \Leftrightarrow \\
\delta(x_1 - x_2) &\sim & K \cdot R \cdot K^T
\end{aligned}
\tag{3}
$$

The aim then is to solve (3) for K. Obviously, this is equivalent to $K^T \cdot K \sim R^{-1}$ (assuming K and R are non-singular), which has a solution $K \sim R^{-\frac{1}{2}}$, unique up to a unitary transformation. The corresponding recurrent filter is then

$$
W = \delta - K^{-1} = \delta - \rho \; R^{\frac{1}{2}}
\tag{4}
$$

This expression suggests an immediate benefit in the use of lateral kernels by the system. As (4) shows, the filter does not now require inverting the correlation matrix and thus is more stable than a feed-forward filter. This also helps preserve the local structure of the autocorrelator in the optimal filter, while, because of the inversion process, a feed-forward system will in general produce non-local, non-topographic solutions.

To obtain a realistic connectivity structure, we need to explicitly include the effects of noise on the system. The system is then described by $O_1 = V + N_1 + M * W * (O_1 + N_2)$, where $N_1$ is the input noise and $N_2$ is the noise, generated by individual units in the recurrently connected layer. Similarly to a feed-forward system (Atick and Redlich 1992), we can modify the decorrelation kernel $W$ derived from (2) to $M * W$. The form of the correction $M$, which minimizes the effects of noise on the system, is obtained by minimizing the distance between the states of the two systems. If we define $\chi^2(M) =< |O - O_1|^2 >$ as the distance function, the solution to $\frac{\partial \chi^2(M)}{\partial M} = 0$ will give us the appropriate kernel. A solution to this problem is

$$
M * W = \delta - (R + N_1^2 + N_2^2) * (\rho \; R^{1/2} + N_2^2)^{-1}
\tag{5}
$$

We see that it has the correct asymptotics as $N_1$, $N_2$ approach zero. The filter behaves well for large $N_2$, turning mostly into a low-pass filter with large attenuation. It cannot handle well large $N_1$ and reaches $-\infty$ proportionally to $N_1^2$.

## 3   Results

### 3.1   Local Optimal Linear Filter

As a first calculation with this model, consider its implications for the connectivity between units in a single hypercolumn. This allows for a very simple application of the theory and does not require any knowledge of the input signal under very general assumptions.

We assume that direction selective cells receive as input from the previous layer the projection of the gradient onto their preferred direction. Thus they act as directional

derivatives, so that their response to a signal with the luminosity profile $L(x)$ and no input from other lateral units is $V_i(x) = |Grad(L(x)).n_i| = |d/dn_i(L(x))|$

With this assumption the outputs of the edge detectors are correlated. Define a (local) correlation matrix $R_{ij} = < V_i V_j >$. By assumption (1), $V_k = |a\, Cos(\alpha - \alpha_k)|$, where $a$ and $\alpha$ are random, independent variables, denoting the magnitude and direction of the local gradient and $\alpha_k$ is the preferred angle of the detector. Assuming spatially isotropic local structure for natural scenes, we can calculate the average of $R$ by integrating over a uniform probability measure in $\alpha$. Then

$$R_{ij} = A \int_0^\pi |Cos(\alpha - \alpha_i)Cos(\alpha - \alpha_j)|\, d\alpha \qquad (6)$$

where $A = < a^2 >$ can be factored because of the assumption of statistical independence. By the homogeneity assumption, $R_{ij}$ is a function of the relative angle $|\alpha_i - \alpha_j|$ only. This allows us to easily calculate the integral in (6) from its Fourier series. Indeed, in Fourier space $\hat{R}$ is just the square of the power spectrum of the underlying signal, i.e., $\cos(\alpha)$ on $[0, \pi]$. Thus we obtain the form of $R$ analytically.

Knowing the local correlations, we can find a recurrent linear filter which decorrelates the outputs after it is applied. This filter is $W = \delta - \rho\, R^{-\frac{1}{2}}$ (Sec.2), unique up to a unitary transformation.

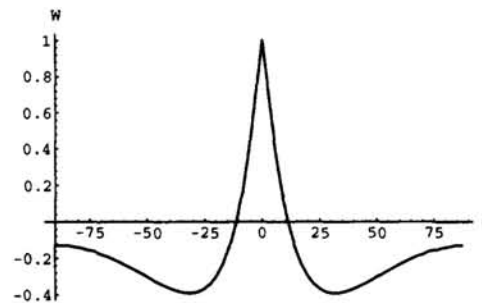

Figure 1: Local recurrent filter in the presence of noise. The connection strength $W$ depends only on the relative angle $\theta$ between units.

If we include noise in the calculation according to (5), we obtain a filter which depends on the signal to noise ratio of the input level. We model the noise process here as a set of independent noise processes for each unit, with $(N_1)_i$ being the input noise and $(N_2)_i$ the output noise for unit $i$. All noise processes are assumed statistically independent. The result for $S/N_2 \sim 3$ is shown on Fig.1. We observe the broadening of the central connections, caused by the need to average local results in order to overcome the noise. It was calculated at very low $N_1$ level, since, as mentioned in Section 2, the filter is unstable with respect to input noise.

With this filter we can directly compare calculations obtained from applying it to a specific input signal, with physiological measurements of the orientation selectivity of cells in the cortex. The results of such comparisons are presented in Fig.2, in which we plot the activity of orientation selective cells in arbitrary units vs stimulus angle in degrees. We see very good matches with experimental results of Celebrini, Thorpe, Trotter, and Imbert (1993), Schiller, Finlay, and Volman (1976) and Orban (1984). We expect some discrepancies, such as in Figures 2.D and 2.F, which can be attributed to the threshold nature of real neural units. We see that we can use the model to classify physiologically distinct cells by the value of the $N_2$ parameter

that describes them. Indeed, since this parameter models the intrinsic noise of a neural unit, we expect it to differ across populations.

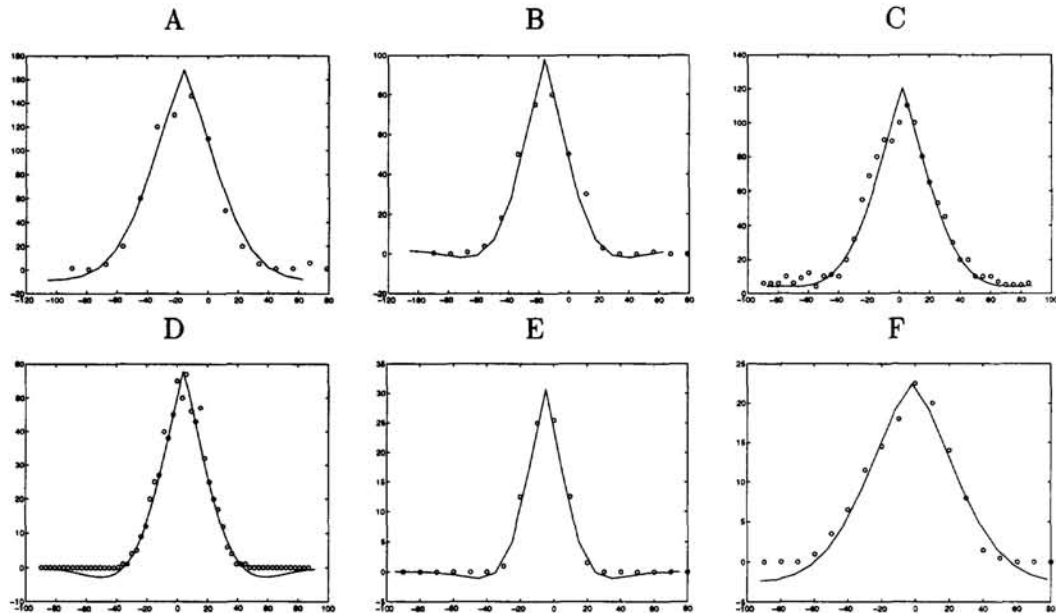

Figure 2: Comparison with experimental data. The activity of orientation selective cells in arbitrary units is plotted against stimulus angle in degrees. Experimental points are denoted with circles, calculated result with a solid line. The variation in the forms of the tuning curves could be accounted for by selecting different noise levels in our noise model. A - data from cell CAJ4 in Celebrini *et.al.* and fit for $N_1 = 0.1, N_2 = 0.2$. B - data from cell CAK2 in Celebrini *et.al.* and fit for $N_1 = 0.35, N_2 = 0.1$. C - data from a complex cell from Orban and fit for $N_1 = 0.3, N_2 = 0.45$. D - data from a simple cell from Orban and fit for $N_1 = 1.0, N_2 = 0.45$. E - data from a simple cells in Schiller *et.al.* and fit for $N_1 = 0.06, N_2 = 0.001$. F - data from a simple cells in Schiller *et.al.* and fit for $N_1 = 15.0, N_2 = 0.01$.

## 3.2   Non-Local Optimal Filter

We can perform a similar analysis of the non-local structure of natural images to design a non-local optimal filter. This time we have a set of detectors $V_k(x) = |a(x) Cos(\alpha(x) - k \pi/N)|$ and a correlation function $R_{ij}(x,y) =< V_i(x) V_j(y) >$, averaged over natural scenes. We assume that the function is spatially translation invariant and can be represented as $R_{ij}(x,y) = R_{ij}(x-y)$. The averaging was done over a set of about 100 different pictures, with 10-20 $256^2$ samples taken from each picture.

The structure of the correlation matrix depends both on the autocorrelator of the gradient field and the structure of the detectors, which are correlated. Obviously the fact that the output units compute $|a(x) Cos(\alpha(x) - k\pi/N)|$ creates many local correlations between neighboring units. Any non-local structure in the detector set is due to a similar structure, present in the gradient field autocorrelator.

The structure of the translation-invariant correlation matrix $R(x)$ is shown in Fig.3A. This can be interpreted as the correlation between the input to the center hypercolumn with the input to rest of the hypercolumns. The result of the complete model (5) can be seen in Fig.3B. Since the filter is also assumed to be translation invariant, the pictures can be interpreted as the connectivity of the center hypercolumn with the rest of the network. This is seen to be concentrated near the diagonal,

A
B

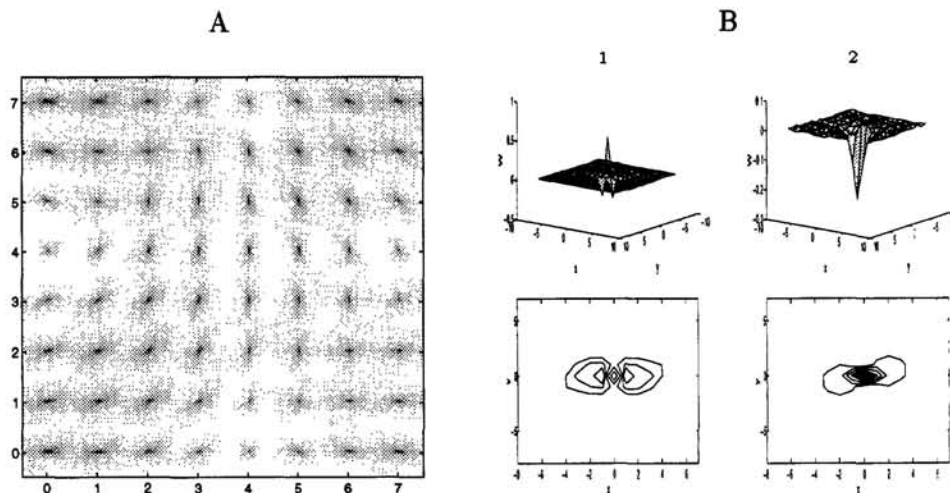

Figure 3: **A.** The autocorrelation function of a set with 8 detectors. Dark represents high correlation, light - low correlation. The sets are indexed by the preferred angles $\theta_i, \theta_j$ in units of $\frac{\pi}{8}$ and each $R_{ij}$ has spatial structure, which is represented as a $32 \times 32$ square. **B.** The lateral connectivity for the central horizontal selective unit with neighboring horizontal (1) and $\pi/4$ (2) selective units. Note the anisotropic connectivity and the rotation of the connectivity axis on the second picture.

and weak in the two adjacent bands, which represent connections to edge detectors with a perpendicular preferred direction. The noise minimizing filter is a low pass filter, as expected, and thus decreases the high frequency component of the power spectrum of the respective decorrelating filter.

## 4  Conclusions and Discussion

We have shown that properties of orientation selective cells in the visual cortex can be partially described by some very simple linear systems analysis. Using this we obtain results which are in very good agreement with physiological and anatomical data of single-cell recordings and imaging. We can use the parameters of the model to classify functionally and structurally differing cells in the visual cortex.

We achieved this by using a recurrent network as the underlying model. This was chosen for several reasons. One is that we tried to give the model biological plausibility and recurrency is well established on the cortical level. Another related heuristic argument is that although there exists a feed-forward network with equivalent properties, as shown in Section 2, such a network will require an additional layer of cells, while the recurrent model allows both for feed-forward processing (the input to our model) as well as manipulation of the output of that (the decorrelation procedure in our model). Finally, while a feed-forward network needs large weights to amplify the signal, a recurrent network is able to achieve very high gains on the input signal with relatively small weights by utilizing special architecture. As can be seen from our equivalence model, $K = (\delta - W)^{-1}$, so if $W$ is so constructed as to have an eigenvalues close to 1, it will produce enormous amplification.

Our work is based on previous suggestions relating the structure of the visual environment to the structure of the visual pathway. It was thought before (Atick and Li 1994) that this particular relation can describe only early visual pathways, but is insufficient to account for the structure of the striate cortex. We show here that redundancy reduction is still sufficient to describe many of the complexities of the visual cortex, thus strengthening the possibility that this is a basic building princi-

ple for the visual system and one should anticipate its appearance in later regions of the latter.

What is even more intriguing is the possibility that this method can account for the structure of other sensory pathways and cortices. We know e.g. that the somatosensory pathway and cortex are similar to the visual ones, because of the similar environments that they encounter (luminosity, edges and textures have analogies in somesthesia). Similar analogies may be expected for the auditory pathway.

We expect even better results if we consider a more realistic non-linear model for the neural units. In fact this improves tremendously the information-processing abilities of a bounded system, since it captures higher order correlations in the signal and allows for true minimization of the mutual information in the system, rather than just decorrelating. Very promising results in this direction have been recently described by Bell and Sejnowski (1996) and Lin and Cowan (1997) and we intend to consider the implications for our model.

## Acknowledgements

Supported in part by Grant # 96–24 from the James S. McDonnell Foundation.

# References

Atick, J. J. and Z. Li (1994). Towards a theory of the striate cortex. *Neural Computation 6*, 127–146.

Atick, J. J. and N. N. Redlich (1992). What does the retina know about natural scenes? *Neural Computation 4*, 196–210.

Atick, J. J. and N. N. Redlich (1993). Convergent algorithm for sensory receptive field developement. *Neural Computation 5*, 45–60.

Attneave, F. (1954). Some informational aspects of visual perception. *Psychological Review 61*, 183–193.

Barlow, H. B. (1989). Unsupervised learning. *Neural Computation 1*, 295–311.

Bell, A. T. and T. J. Sejnowski (1996). The "independent components" of natural scences are edge filters. *Vision Research* (submitted).

Celebrini, S., S. Thorpe, Y. Trotter, and M. Imbert (1993). Dynamics of orientation coding in area V1 of the awake primate. *Visual Neuroscience 10*, 811–825.

Dong, D. (1994). Associative decorrelation dynamics: a theory of self-organization and optimization in feedback networks. Volume 7 of *Advances in Neural Information Processing Systems*, pp. 925–932. The MIT Press.

Field, D. J. (1987). Relations between the statistics of natural images and the response properties of cortical cells. *J. Opt. Soc. Am. 4*, 2379–2394.

Lin, J. K. and J. D. Cowan (1997). Faithful representation of separable input distributions. *Neural Computation*, (to appear).

Orban, G. A. (1984). *Neuronal Operations in the Visual Cortex*. Springer-Verlag, Berlin.

Ruderman, D. L. and W. Bialek (1993). Statistics of natural images: Scaling in the woods. In J. D. Cowan, G. Tesauro, and J. Alspector (Eds.), *Advances in Neural Information Processing Systems*, Volume 6. Morgan Kaufman, San Mateo, CA.

Schiller, P., B. Finlay, and S. Volman (1976). Quantitative studies of single-cell properties in monkey striate cortex. II. Orientation specificity and ocular dominance. *J. Neuroph. 39*(6), 1320–1333.